# Acquisition in Autoshaping

**Sham Kakade**          **Peter Dayan**
Gatsby Computational Neuroscience Unit
17 Queen Square, London, England, WC1N 3AR.
sham@gatsby.ucl.ac.uk    dayan@gatsby.ucl.ac.uk

## Abstract

Quantitative data on the speed with which animals acquire behavioral responses during classical conditioning experiments should provide strong constraints on models of learning. However, most models have simply ignored these data; the few that have attempted to address them have failed by at least an order of magnitude. We discuss key data on the speed of acquisition, and show how to account for them using a statistically sound model of learning, in which differential *reliabilities* of stimuli play a crucial role.

## 1 Introduction

Conditioning experiments probe the ways that animals make predictions about rewards and punishments and how those predictions are used to their advantage. Substantial quantitative data are available as to how pigeons and rats *acquire* conditioned responses during *autoshaping*, which is one of the simplest paradigms of classical conditioning.[4] These data are revealing about the statistical, and ultimately also the neural, substrate underlying the ways that animals learn about the causal texture of their environments.

In autoshaping experiments on pigeons, the birds acquire a peck response to a lighted key associated (irrespective of their actions) with the delivery of food. One attractive feature of autoshaping is that there is no need for separate 'probe trials' to assess the degree of association formed between the light and the food by the animal — rather, the rate of key pecking during the light (and before the food) can be used as a direct measure of this association. In particular, acquisition speeds are often measured by the number of trials until a certain behavioral criterion is met, such as pecking during the light on three out of four successive trials.[4,8,10]

As stressed persuasively by Gallistel & Gibbon[4] (GG; forthcoming), the critical feature of autoshaping is that there is substantial experimental evidence on how acquisition speed depends on the three critical variables shown in figure 1A. The first is $I$, the inter-trial interval; the second is $T$, the time during the trial for which the light is presented; the third is the training schedule, $1/S$, which is the fractional number of deliveries per light — some birds were only partially reinforced.

Figure 1 makes three key points. First, figure 1B shows that the median number of trials to the acquisition criterion depends on the *ratio* of $I/T$, and not on $I$ and $T$ separately – experiments reported for the same $I/T$ are actually performed with $I$ and $T$ differing by more than an order of magnitude.[4,8] Second, figure 1B shows convincingly that the number of reinforcements is approximately inversely proportional to $I/T$ — the relatively shorter presentation of light, the faster the learn-

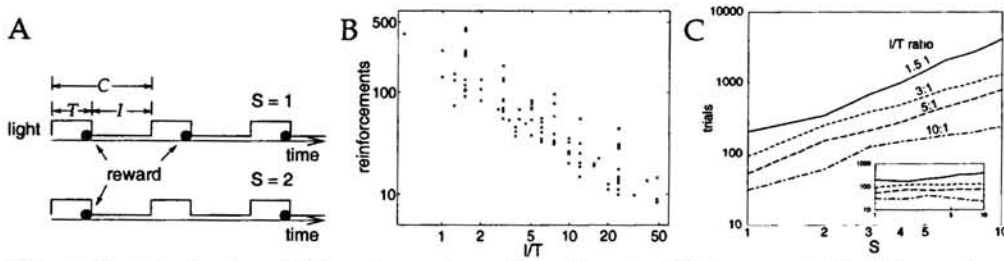

**Figure 1:** Autoshaping. A) Experimental paradigm. Top: the light is presented for $T$ seconds every $C$ seconds and is always followed by the delivery of food (filled circle). Bottom: the food is delivered with probability $1/S = 1/2$ per trial. In some cases $I$ is stochastic, with the appropriate mean. B) Log-log plot[4] of the number of reinforcements to a given acquisition criterion versus the $I/T$ ratio for $S = 1$. The data are median acquisition times from 12 different laboratories. C) Log-log acquisition curves for various $I/T$ ratios and $S$ values. The main graph shows *trials versus S*; the inset shows *reinforcements versus S*. (1999).

ing. Third, figure 1C shows that partial reinforcement has almost no effect when measured as a function of the number of reinforcements (rather than the number of trials),[4, 10] since although it takes $S$ times as many *trials* to acquire, there are *reinforcements* on only $1/S$ trials. Changing $S$ does not change the effective $I/T$ when measured as a function of reinforcements, so this result might actually be expected on the basis of figure 1B, and we only consider $S = 1$ in this paper. Altogether, the data show that:

$$n \approx 300 * T/I \qquad (1)$$

where $n$ is the number of rewards to the acquisition criterion. Remarkably, these effects seem to hold for over an order of magnitude in both $I/T$ and $S$.

These quantitative data should be a most seductive target for statistically sound models of learning. However, few models have even attempted to capture the strong constraints they provide, and those that have attempted, all fail in critical aspects. The best of them, rate estimation theory[4] (RET), is closely related to the Rescorla-Wagner[13] (RW) model, and actually captures the proportionality in equation 1. However, as shown below, RET grossly overestimates the observed speed of acquisition (underestimating the proportionality constant). Further, RET is designed to account for the time at which a particular, standard, acquisition criterion is met. Figure 2A shows that this is revealing only about the very early stages of learning — RET is silent about the remainder of the learning curve.

We look at additional quantitative data on learning, which collectively suggest that stimuli *compete* to predict the delivery of reward. Dayan & Long[3] (DL) discussed various statistically inspired competitive models of classical conditioning, concluding with one in which stimuli are differently *reliable* as predictors of reward. However, DL ignored the data shown in figures 1 and 2, basing their analysis on conditioning paradigms in which $I/T$ was not a factor. Figures 1 and 2 demand a more sophisticated statistical model — building such a model is the focus of this paper.

## 2 Rate Estimation Theory

Gallistel & Gibbon[4] (GG; forthcoming) are amongst the strongest proponents of the quantitative relationships in figure 1. To account for them, GG suggest that animals are estimating the rates of rewards — one, $\lambda_l$, for the rate associated with the light and another, $\lambda_b$, for the rate associated with the *background context*. The context is the ever-present environment which can itself gain associative value. The overall

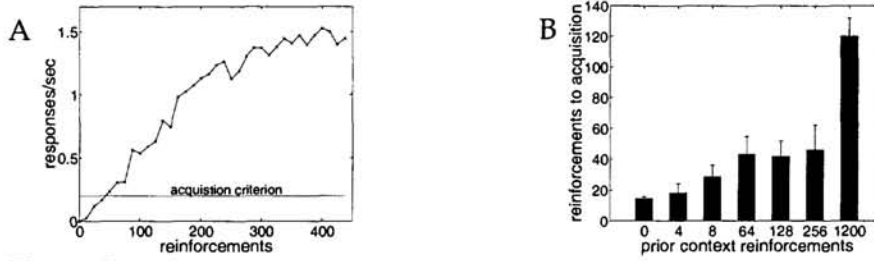

Figure 2: Additional Autoshaping Data. A) Acquisition of keypecking. The figure shows response rate *versus* reinforcements.[6] The acquisition criterion is satisfied at a relatively *early* time when the response curve crosses the acquisition criterion line. B) The effects of prior context reinforcements on subsequent acquisition speed. The data are taken from two experiments,[1,2] with $I/T = 6$.

predicted reward rate while the light is on is $\lambda_l + \lambda_b$, and the rate without the light is just $\lambda_b$.

The additive form of the model makes it similar to Rescorla-Wagner's[13] (RW) standard delta-rule model, for which the net prediction of the expected reward in a trial is the sum of the associative values of each active predictor (in this case, the context and light). If the rewards are modeled as being just present or absent, the expected value for a reward is just its *probability* of occurrence. Instead, RET uses *rates*, which are just probabilities per unit time.

GG[4] formulated their model from a frequentist viewpoint. However, it is easier to discuss a closely related Bayesian model which suffers from the same underlying problem. Instead of using RW's delta-rule for learning the rates, GG assume that reinforcements come from a constant rate Poisson process, and make sound statistical inferences about the rates given the data on the rewards. Using an improper flat prior over the rates, we can write the joint distribution as:

$$\mathcal{P}(\lambda_l \lambda_b \mid \text{data}) \propto \mathcal{P}(n \mid \lambda_l \lambda_b t_l t_b) \propto (\lambda_l + \lambda_b)^n e^{-(\lambda_l + \lambda_b)t_l} e^{-\lambda_b t_b} \qquad (2)$$

since all $n$ rewards occur with the light, at rate $\lambda_l + \lambda_b$. Here, $t_l = nT$ is the total time the light is on, and $t_b = nI$ is the total time the light is off.

GG take the further important step of relating the inferred rates $\lambda_l$ and $\lambda_b$ to the decision of the animals to start responding (*ie* to satisfy the acquisition criterion). GG suggest that acquisition should occur when the animals have strong evidence that the fractional increase in the reward rate, whilst the light is on, is greater than some threshold. More formally, acquisition should occur when:

$$\mathcal{P}((\lambda_l + \lambda_b)/\lambda_b > \beta \mid n) = 1 - \alpha \qquad (3)$$

where $\alpha$ is the uncertainty threshold and $\beta$ is slightly greater than one, reflecting the fractional increase. The $n$ that first satisfies equation 3 can be found by integrating the joint probability in equation 2. It turns out that $n \propto t_l/t_b$, which has the approximate, linear dependence on the ratio $I/T$ (as in figure 1B), since $t_l/t_b = nT/nI = T/I$. It also has no dependence on partial reinforcement, as observed in figure 1C.

However, even with a very low uncertainty, $\alpha = 0.001$, and a reasonable fractional increase, $\beta = 1.5$, this model predicts that learning should be more than ten times as fast as observed, since we get $n \approx 20 * T/I$ as opposed to the $300 * T/I$ observed. Equation 1 can only be satisfied by setting $\alpha$ between $10^{-20}$ and $10^{-50}$ (depending on the precise values of $I/T$ and $\beta$)! This spells problems for GG as a normative, ideal detector model of learning — it cannot, for instance, be repaired with any reasonable prior for the rates, as $\alpha$ drops drastically with $n$. In other circumstances,

though, Gallistel, Mark & King[5] (forthcoming) have shown that animals can be ideal detectors of changes in rates.

One hint of the flaw with GG is that simple manipulations to the context before starting autoshaping (in particular *extinction*) can produce very *rapid* learning.[2] More generally, the data show that acquisition speed is strongly controlled by prior rewards being given only in the context (without the light present).[2] Figure 2B shows a parametric study of subsequent acquisition speeds during autoshaping as a function of the *number* of rewards given only with the context. This effect cannot simply be modeled by assuming a different prior distribution for the rates (which does not fix the problem of the speed of acquisition in any case), since the *rate* at which these prior context rewards were given has little effect on subsequent acquisition speed for a given *number* of prior reinforcements.[9] Note that the data in figure 2B (*ie* equation 1) suggest that there were about thirty prior rewards in the context — this is consistent with the experimental procedures used,[8–10] although prior experience was not a carefully controlled factor.

## 3   The Competitive Model

Five sets of constraints govern our new model. First, since animals can be ideal detectors of rates in some circumstances,[5] we only consider accounts under which their acquisition of responding has a rational statistical basis. Second, the number of reinforcements to acquisition must be $n \approx 300 * T/I$, as in equation 1. This requires that the constant of proportionality should come from rational, not absurd, uncertainties. Third, pecking rates after the acquisition criterion is satisfied should also follow the form of figure 2A (in the end, we are preventing from a normative account of this by a dearth of data). Fourth, the overall learning speed should be strongly affected by the *number* of prior context rewards (figure 2B), but not by the *rate* at which they were presented. That is, the context, as an established predictor, regardless of the rate it predicts, should be able to substantially *block* learning to a less established predictor. Finally, the asymptotic accuracy of rate estimates should satisfy the substantial experimental data on the intrinsic uncertainty in the predictions in the form of a quantitative account called scalar expectancy theory[7] (SET).

In our model, as in DL, an *independent* prediction of the rate of reward delivery is made on the basis of each stimulus that is present ($\omega_c$, for the context; $\omega_l$ for the light). These separate predictions are combined based on estimated *reliabilities* of the predictions. Here, we present a heuristic version of a more rigorously specified model.[12]

### 3.1   Rate Predictions

SET[7] was originally developed to capture the nature of uncertainty in the way that animals estimate time intervals. Its most important result is that the standard deviation of an estimate is consistently proportional to the mean, even after an asymptotic number of presentations of the interval. Since the estimated time to a reward is just the inverse rate, asymptotic rate estimates might also be expected to have constant coefficients of variation. Therefore, we constrain the standard deviations of rate estimates not to drop below a multiple of their means. Evidence suggests that this multiple is about 0.2.[7] RET clearly does not satisfy this constraint as the joint distribution (equation 2) becomes arbitrarily accurate over time.

Inspired by Sutton,[14] we consider Kalman filter models for *independent* log-predictions, $\log \omega_c(m)$ and $\log \omega_l(m)$, on trial m. The output models for the filters

specify the relationship between the predicted and observed rates. We use a simple log-normal, $\mathcal{LN}$, approximation (to an underlying truly Poisson model):

$$P(o_c(m) \mid \omega_c(m)) \sim \mathcal{LN}(\omega_c(m), v_c^2) \quad P(o_l(m) \mid \omega_l(m)) \sim \mathcal{LN}(\omega_l(m), v_l^2) \quad (4)$$

where $o_*(m)$ is the observed average reward whilst predictor $*$ is present, so if a reward occurs with the light in trial $m$, then $o_l(m) = 1/T$ and $o_c(m) = 1/C$ (where $C = T + I$). The values of $v_*^2$ can be determined, from the Poisson model, to be $v_c^2 = v_l^2 = 1$.

The other part of the Kalman filter is a model of change in the world for the $\omega$'s:

$$\log \omega_c(m) = \log \omega_c(m-1) + \epsilon_c(m) \qquad \epsilon_c(m) \sim \mathcal{N}(0, (\eta(\eta+1))^{-1}) \quad (5)$$

$$\log \omega_l(m) = \log \omega_l(m-1) + \epsilon_l(m) \qquad \epsilon_l(m) \sim \mathcal{N}(0, (\eta(\eta+1))^{-1}) \quad (6)$$

We use log(rates) so that there is no inherent scale to change in the world. Here, $\eta$ is a constant chosen to satisfy the SET constraint, imposed as $\sigma_* = \widehat{\omega}_*/\sqrt{\eta}$ at asymptote. Notice that $\eta$ acts as the effective number of rewards remembered, which will be less than 30, to get the observed coefficient of variation above 0.2.

After observing the data from $m$ trials, the posterior distributions for the predictions will become approximately:

$$P(\omega_c(m) \mid \text{data}) \sim \mathcal{N}(1/C, \sigma_c^2(m)) \quad P(\omega_l(m) \mid \text{data}) \sim \mathcal{N}(1/T, \sigma_l^2(m)) \quad (7)$$

and, in about $m = \eta$ trials, $\sigma_c(m) \to (1/C)/\sqrt{\eta}$ and $\sigma_l(m) \to (1/T)/\sqrt{\eta}$. This captures the fastest acquisition in figure 2, and also extinction.

### 3.2 Cooperative Mixture of Experts

The two predictions (equation 7) are combined using the factorial experts model of Jacobs *et al*[11] that was also used by DL. For this, *during* the presentation of the light (and the context, of course), we consider that, independently, the relationships between the actual reward rate $r(m)$ and the outputs $\omega_l(m)$ and $\omega_c(m)$ of 'experts' associated with each stimulus are:

$$P(\omega_l(m)|r(m)) \sim \mathcal{N}(r(m), \tfrac{1}{\rho_l(m)}) \quad , \quad P(\omega_c(m)|r(m)) \sim \mathcal{N}(r(m), \tfrac{1}{\rho_c(m)}) \quad (8)$$

where $\rho_l(m)^{-1}$ and $\rho_c(m)^{-1}$ are inverse variances, or *reliabilities* for the stimuli. These reliabilities reflect the belief as to how close $\omega_l(m)$ and $\omega_c(m)$ are to $r(m)$. The estimates are combined, giving

$$P(r(m) \mid \omega_l(m), \omega_c(m)) \sim \mathcal{N}(\widehat{r}(m), (\rho_l(m) + \rho_c(m))^{-1})$$

$$\widehat{r}(m) = \pi_l(m)\omega_l(m) + (1 - \pi_l(m))\omega_c(m) \quad \pi_l(m) = \rho_l(m)/(\rho_l(m) + \rho_c(m))$$

The prediction of the reward rate without the light $r_c(m)$ is determined just by the context value $\omega_c(m)$.

In this formulation, the context can block the light's prediction if it is more *reliable* ($\rho_c \gg \rho_l$), since $\pi_l \approx 0$, making the mean $\widehat{r}(m) \approx \omega_c(m)$, and this blocking occurs regardless of the context's *rate*, $\omega_c(m)$. If $\rho_l$ slowly increases, then $\widehat{r}(m) \to \omega_l$ slowly as $\pi_l(m) \to 1$. We expect this to model the post-acquisition part of the learning shown in figure 2A.

A fully normative model of acquisition would come from a statistically correct account of how the reliabilities should change over time, which, in turn, would come from a statistical model of the expectations the animal has of how predictabilities change in the world. Unfortunately, the slow phase of learning in figure 2A, which should provide the most useful data on these expectations, is almost ubiquitously

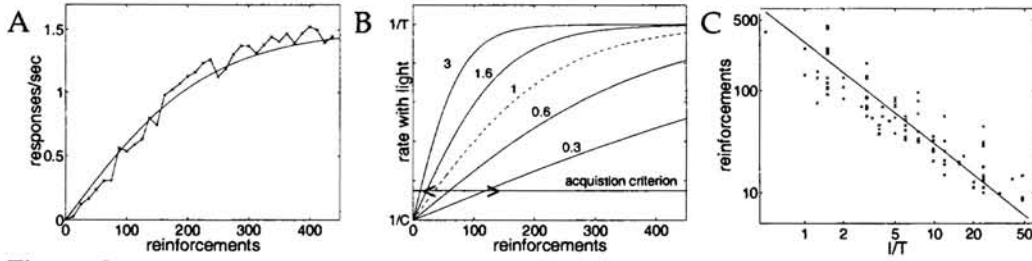

Figure 3: Satisfaction of the Constraints. A) The fit to the behavioral response curve (figure 2B), using equation 9 and $\pi_0 = 0.004$. B) Possible acquisition curves showing $\widehat{r}(m)$ *versus* $m$. The $\longleftrightarrow$ on the criterion line denotes the range of 15 to 120 reinforcements that are indicated by figure 2B. The $--$ curve is the same as in Fig 3A. The parameters displayed are values for $\pi_0$ in multiples of $\pi_0$ for the center curve. C) A theoretical fit to the data using equation 11. Here, $\alpha = 5\%$ and $\pi_0\sqrt{\rho_0} = 0.004$.

ignored in experiments. We therefore make two assumptions about this, which are chosen to fit the acquisition data, but whose normative underpinnings are unclear. The first assumption, chosen to obtain the slow learning curve, is that:

$$\pi_l(m) = \tanh \pi_0 m \tag{9}$$

Assuming that the strength of the behavioral response is approximately proportional to $r(m) - r_c(m)$, which we will estimate by $\pi_l(m)(\widehat{\omega}_l(m) - \widehat{\omega}_c(m))$, figure 3A compares the rate of key pecking in the model with the data from figure 2A. Figure 3B shows the effect on the behavioral response of varying $\pi_0$. Within just a half an order magnitude of variation of $\pi_0$, the acquisition speeds (judged at the criterion line shown) due to between 1200 and 0 prior context rewards (figure 2B) can be obtained. Note the slightly counter-intuitive explanation — the actual reward rate associated with the light is established very quickly — slow learning comes from slow changes in the importance paid to these rates.

We make a second assumption that the coefficient of variation of the context's prediction, from equation 8, does not change significantly for the early trials before the acquisition criterion is met (it could change thereafter). This gives:

$$\rho_c(m) \approx \rho_0/\widehat{\omega}_c(m)^2 \quad \text{for early } m \tag{10}$$

It is plausible that the context is not becoming a relatively worse 'expert' for early $m$, since no other predictor has yet proven more reliable.

Following GG's suggestion, we model acquisition as occurring on trial $m$ if $\mathcal{P}(r(m) > r_c(m)|\text{data}) \geq 1 - \alpha$, *ie* if the animal has sound reasons to expect a higher reward rate with the light. Integrating over the Kalman filter distributions in equation 7 gives the distribution of $r(m) - r_c(m)$ for early $m$ as

$$\mathcal{P}(r(m) - r_c(m) \mid \text{data}) \sim \mathcal{N}((\tanh \pi_0 m)(1/T - 1/C), (\rho_0 C^2)^{-1})$$

where $\sigma_*(m)$ has dropped out due to $\pi_l(m)$ being small at early $m$. Finding the number of rewards, $n$, that satisfies the acquisition criterion gives:

$$n \approx \frac{\alpha}{\pi_0\sqrt{\rho_0}}\frac{T}{I} \tag{11}$$

where the factor of $\alpha$ depends on the uncertainty, $\alpha$, used. Figure 3C shows the theoretical fit to the data.

## 4  Discussion

Although a noble attempt, RET fails to satisfy the strong body of constraints under which any acquisition model must labor. Under RET, the acquisition of responding cannot have a rational statistical basis, as the animal's modeled uncertainty in

the association between light and reward at the time of acquisition is below $10^{-20}$. Further, RET ignores constraints set forth by the data establishing SET and also data on prior context manipulations. These latter data show that the context, regardless of the rate it predicts, will substantially block learning to a less established predictor. Additive models, such as RET, are unable to capture this effect.

We have suggested a model in which each stimulus is like an 'expert' that learns independently about the world. Expert predictions can adapt quickly to changes in contingencies, as they are based on a Kalman filter model, with variances chosen to satisfy the constraint suggested by SET, and they can be combined based on their *reliabilities*. We have demonstrated the model's close fit to substantial experimental data. In particular, the new model captures the $I/T$ dependence of the number of rewards to acquisition, with a constant of proportionality that reflects rational statistical beliefs. The slow learning that occurs in some circumstances, is due to a slow change in the reliabilities of predictors, not due to the rates being unable to adapt quickly. Although we have not shown it here, the model is also able to account for quantitative data as to the speed of extinction of the association between the light and the reward.

The model leaves many directions for future study. In particular, we have not specified a sound statistical basis for the changes in reliabilities given in equations 9 and 10. Such a basis is key to understanding the slow phase of learning. Second, we have not addressed data from more sophisticated conditioning paradigms. For instance, overshadowing, in which multiple conditioned stimuli are similarly predictive of the reward, should be able to be incorporated into the model in a natural way.

## Acknowledgements

We are most grateful to Randy Gallistel and John Gibbon for freely sharing, prior to publication, their many ideas about timing and conditioning. We thank Sam Roweis for comments on an earlier version of the manuscript. Funding is from a NSF Graduate Research Fellowship (SK) and the Gatsby Charitable Foundation.

## References

[1] Balsam, PD, & Gibbon, J (1988). *Journal of Experimental Psychology: Animal Behavior Processes*, **14**: 401-412.

[2] Balsam, PD, & Schwartz, AL (1981). *Journal of Experimental Psychology: Animal Behavior Processes*, **7**: 382-393.

[3] Dayan, P, & Long, T, (1997) *Neural Information Processing Systems*, **10**:117-124.

[4] Gallistel, CR, & Gibbon, J (1999). *Time, Rate, and Conditioning*. Forthcoming.

[5] Gallistel, CR, Mark, TS & King, A (1999). *Is the Rat an Ideal Detector of Changes in Rates of Reward?*. Forthcoming.

[6] Gamzu, ER, & Williams, DR (1973). *Journal of the Experimental Analysis of Behavior*, **19**:225-232.

[7] Gibbon, J (1977). *Psychological Review* **84**:279-325.

[8] Gibbon, J, Baldock, MD, Locurto, C, Gold, L & Terrace, HS (1977). *Journal of Experimental Psychology: Animal Behavior Processes*, **3**: 264-284.

[9] Gibbon, J & Balsam, P (1981). In CM Locurto, HS Terrace, & J Gibbon, editors, *Autoshaping and Conditioning Theory*. 219-253. New York, NY: Academic Press.

[10] Gibbon, J, Farrell, L, Locurto, CM, Duncan, JH & Terrace, HS (1980). *Animal Learning and Behavior*, **8**:45-59.

[11] Jacobs, RA, Jordan, MI, & Barto, AG (1991). *Cognitive Science* **15**:219-250.

[12] Kakade, S & Dayan, P (2000). In preparation.

[13] Rescorla, RA & Wagner, AR (1972). In AH Black & WF Prokasy, editors, *Classical Conditioning II: Current Research and Theory*, 64-69. New York, NY: Appleton-Century-Crofts.

[14] Sutton, R (1992). In *Proceedings of the 7th Yale Workshop on Adaptive and Learning Systems*.
